# Combined Neural Network and Rule-Based Framework for Probabilistic Pattern Recognition and Discovery

**Hayit K. Greenspan and Rodney Goodman**
Department of Electrical Engineering
California Institute of Technology, 116-81
Pasadena, CA 91125

**Rama Chellappa**
Department of Electrical Engineering
Institute for Advanced Computer Studies and Center for Automation Research
University of Maryland, College Park, MD 20742

## Abstract

A combined neural network and rule-based approach is suggested as a general framework for pattern recognition. This approach enables unsupervised and supervised learning, respectively, while providing probability estimates for the output classes. The probability maps are utilized for higher level analysis such as a feedback for smoothing over the output label maps and the identification of unknown patterns (pattern "discovery"). The suggested approach is presented and demonstrated in the texture - analysis task. A correct classification rate in the 90 percentile is achieved for both unstructured and structured natural texture mosaics. The advantages of the probabilistic approach to pattern analysis are demonstrated.

## 1 INTRODUCTION

In this work we extend a recently suggested framework (Greenspan et al,1991) for a combined neural network and rule-based approach to pattern recognition. This approach enables unsupervised and supervised learning, respectively, as presented in Fig. 1. In the unsupervised learning phase a neural network clustering scheme is used for the quantization of the input features. A supervised stage follows in which labeling of the quantized attributes is achieved using a rule based system. This information theoretic technique is utilized to find the most informative correlations between the attributes and the pattern class specification, while providing probability estimates for the output classes. Ultimately, a minimal representation for a library of patterns is learned in a training mode, following which the classification of new patterns is achieved.

The suggested approach is presented and demonstrated in the texture - analysis task. Recent results (Greenspan et al, 1991) have demonstrated a correct classification rate of 95 - 99% for synthetic (texton) textures and in the 90 percentile for 2 - 3 class natural texture mosaics. In this paper we utilize the output probability maps for high-level analysis in the pattern recognition process. A feedback based on the confidence measures associated with each class enables a smoothing operation over the output maps to achieve a high degree of classification in more difficult (natural texture) pattern mosaics. In addition, a generalization of the recognition process to identify unknown classes (pattern "discovery"), in itself a most challenging task, is demonstrated.

## 2   FEATURE EXTRACTION STAGE

The initial stage for a classification system is the feature extraction phase through which the attributes of the input domain are extracted and presented towards further processing. The chosen attributes are to form a representation of the input domain, which encompasses information for any desired future task.

In the texture-analysis task there is both biological and computational evidence supporting the use of Gabor filters for the feature - extraction phase (Malik and Perona, 1990; Bovik et al, 1990). Gabor functions are complex sinusoidal gratings modulated by 2-D Gaussian functions in the space domain, and shifted Gaussians in the frequency domain. The 2-D Gabor filters form a complete but non-orthogonal basis which can be used for image encoding into multiple spatial frequency and orientation channels. The Gabor filters are appropriate for textural analysis as they have tunable orientation and radial frequency bandwidths, tunable center frequencies, and optimally achieve joint resolution in space and spatial frequency.

In this work, we use the Log Gabor pyramid, or the Gabor wavelet decomposition to define an initial finite set of filters. We implement a pyramidal approach in the filtering stage reminiscent of the Laplacian Pyramid (Burt and Adelson, 1983). In our simulations a computationally efficient scheme involves a pyramidal representation of the image which is convolved with fixed spatial support oriented Gabor filters. Three scales are used with 4 orientations per scale (0,90,45,-45 degrees), together with a non-oriented component, to produce a 15-dimensional feature vector for every local window in the original image, as the output of the feature extraction stage.

The pyramidal approach allows for a hierarchical, multiscale framework for the image analysis. This is a desirable property as it enables the identification of features at various scales of the image and thus is attractive for scale-invariant pattern

recognition.

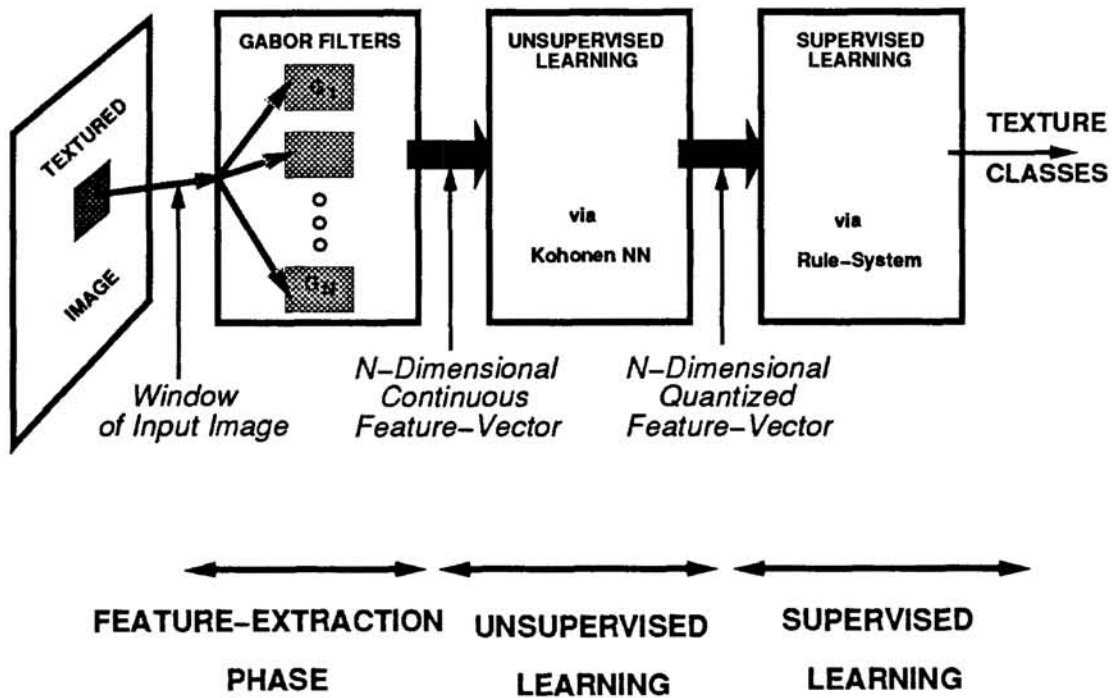

Figure 1: System Block Diagram

# 3 QUANTIZATION VIA UNSUPERVISED LEARNING

The unsupervised learning phase can be viewed as a preprocessing stage for achieving yet another, more compact representation, of the filtered input. The goal is to quantize the continuous valued features which are the result of the initial filtering stage. The need for discretization becomes evident when trying to learn associations between attributes in a statistically-based framework, such as a rule-based system. Moreover, in an extended framework, the network can reduce the dimension of the feature domain. This shift in representation is in accordance with biological based models.

The output of the filtering stage consists of $N$ (=15) continuous valued feature maps; each representing a filtered version of the original input. Thus, each local area of the input image is represented via an $N$-dimensional feature vector. An array of such $N$-dimensional vectors, viewed across the input image, is the input to the learning stage. We wish to detect characteristic behavior across the $N$-dimensional feature space for the family of textures to be learned. By projecting an input set of samples onto the $N$-dimensional space, we search for clusters to be related to corresponding code-vectors, and later on, recognized as possible texture classes. A neural-network quantization procedure, based on Kohonen's model (Kohonen, 1984) is utilized for this stage.

In this work each dimension, out of the $N$-dimensional attribute vector, is individually clustered. All samples are thus projected onto each axis of the space and

one-dimensional clusters are found; this scalar quantization case closely resembles the K-means clustering algorithm. The output of the preprocessing stage is an $N$-dimensional quantized vector of attributes which is the result of concatenating the discrete valued codewords of the individual dimensions. Each dimension can be seen to contribute a probabilistic differentiation onto the different classes via the clusters found. As some of the dimensions are more representative than others, it is the goal of the supervised stage to find the most informative dimensions for the desired task (with the higher differentiation capability ) and to label the combined clustered domain.

## 4   SUPERVISED LEARNING VIA A RULE-BASED SYSTEM

In the supervised stage we utilize the existing information in the feature maps for higher level analysis, such as input labeling and classification. In particular we need to learn a classifier which maps the output attributes of the unsupervised stage to the texture class labels. Any classification scheme could be used. However, we utilize a rule - based information theoretic approach which is an extension of a first order Bayesian classifier, because of its ability to output probability estimates for the output classes (Goodman et al, 1992). The classifier defines correlations between input features and output classes as probabilistic rules of the form: If $Y = y$ then $X = x$ with prob. $p$, where $Y$ represents the attribute vector and $X$ is the class variable. A data driven supervised learning approach utilizes an information theoretic measure to learn the most informative links or rules between the attributes and the class labels. Such a measure was introduced as the $J$ measure (Smyth and Goodman, 1991) which represents the information content of a rule as the average bits of information that attribute values $y$ give about the class $X$. The most informative set of rules via the $J$ measure is learned in a training stage, following which the classifier uses them to provide an estimate of the probability of a given class being true. When presented with a new input evidence vector, $Y$, a set of rules can be considered to "fire". The classifier estimates the log posterior probability of each class given the rules that fire as:

$$\log p(x | \textit{rules that fire}) = \log p(x) + \sum_j W_j$$

$$W_j = \log \left( \frac{p(x|y)}{p(x)} \right)$$

where $p(x)$ is the prior probability of the class $x$, and $W_j$ represents the evidential support for the class as provided by rule $j$. Each class estimate can now be computed by accumulating the "weights of evidence" incident it from the rules that fire. The largest estimate is chosen as the initial class label decision. The probability estimates for the output classes can now be used for feedback purposes and further higher level processing.

The rule-based classification system can be mapped into a 3 layer feed forward architecture as shown in Fig. 2. The input layer contains a node for each attribute.

The hidden layer contains a node for each rule and the output layer contains a node for each class. Each rule (second layer node $j$) is connected to a class via the multiplicative weight of evidence $W_j$.

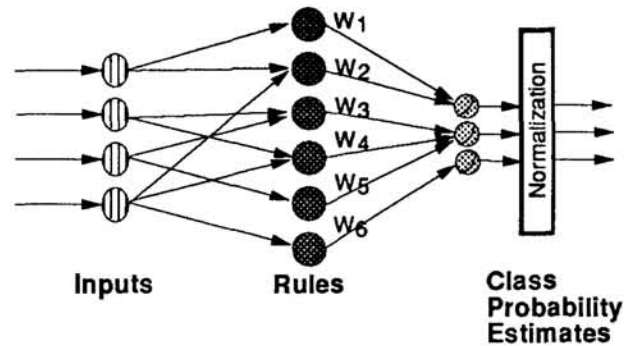

Figure 2: Rule-Based Network

## 5    RESULTS

In previous results (Greenspan et al, 1991) we have shown the capability of the proposed system to recognize successfully both artificial ("texton") and natural textures. A classification rate of 95-99% was obtained for 2 and 3 class artificial images. 90-98% was achieved for 2 and 3 class natural texture mosaics. In this work we wish to demonstrate the advantage of utilizing the output probability maps in the pattern recognition process. The probability maps are utilized for higher level analysis such as a feedback for smoothing and the identification of unknown patterns (pattern "discovery"). An example of a 5 - class natural texture classification is presented in Fig. 3. The mosaic is comprised of grass, raffia, herring, wood and wool (center square) textures. The input mosaic is presented (top left), followed by the labeled output map (top right) and the corresponding probability maps for a prelearned library of 6 textures (grass, raffia, wood, calf, herring and wool, left to right, top to bottom, respectively). The input poses a very difficult task which is challenging even to our own visual perception. Based on the probability maps (with white indicating strong probability) the very satisfying result of the labeled output map is achieved. The 5 different regions have been identified and labeled correctly (in different shades of gray) with the boundaries between the regions very strongly evident. A feedback based on the probability maps was used for smoothing over the label map, to achieve the result presented. It is worthwhile noting that the probabilistic framework enables the analysis of both structural textures (such as the wood, raffia and herring) and unstructural textures (such as the grass and wool).

Fig. 4. demonstrates the generalization capability of our system to the identification of an unknown class. In this task a presented pattern which is not part of the prelearned library, is to be recognized as such and labeled as an unknown area of interest. This task is termed "pattern discovery" and its application is wide spread from identifying unexpected events to the presentation of areas-of-interest in scene exploratory studies. Learning the unknown is a difficult problem in which the probability estimates prove to be valuable. In the presented example a 3 texture library

was learned, consisting of wood, raffia and grass textures. The input consists of wood, raffia and sand (top left). The output label map (top right) which is the result of the analysis of the respective probability maps (bottom) exhibits the accurate detection of the known raffia and wood textures, with the sand area labeled in black as an unknown class. This conclusion was based on the corresponding probability estimations which are zeroed out in this area for all the known classes. We have thus successfuly analyzed the scene based on the existing source of knowledge.

Our most recent results pretain to the application of the system to natural scenery analysis. This is a most challanging task as it relates to real-world applications, an example of which are NASA space exploratory goals. Initial simulation results are presented in Fig. 5. which presents a sand-rock scenerio. The training examples are presented, followed by two input images and their corresponding output label maps, left to right, respectively. Here, white represents rock, gray represents sand and black regions are classified as unknown. The system copes successfully with this challange. We can see that a distinction between the regions has been made and for a possible mission such as rock avoidence (landing purposes, navigation etc.) reliable results were achieved. These initial results are very encouraging and indicate the robustness of the system to cope with difficult real-world cases.

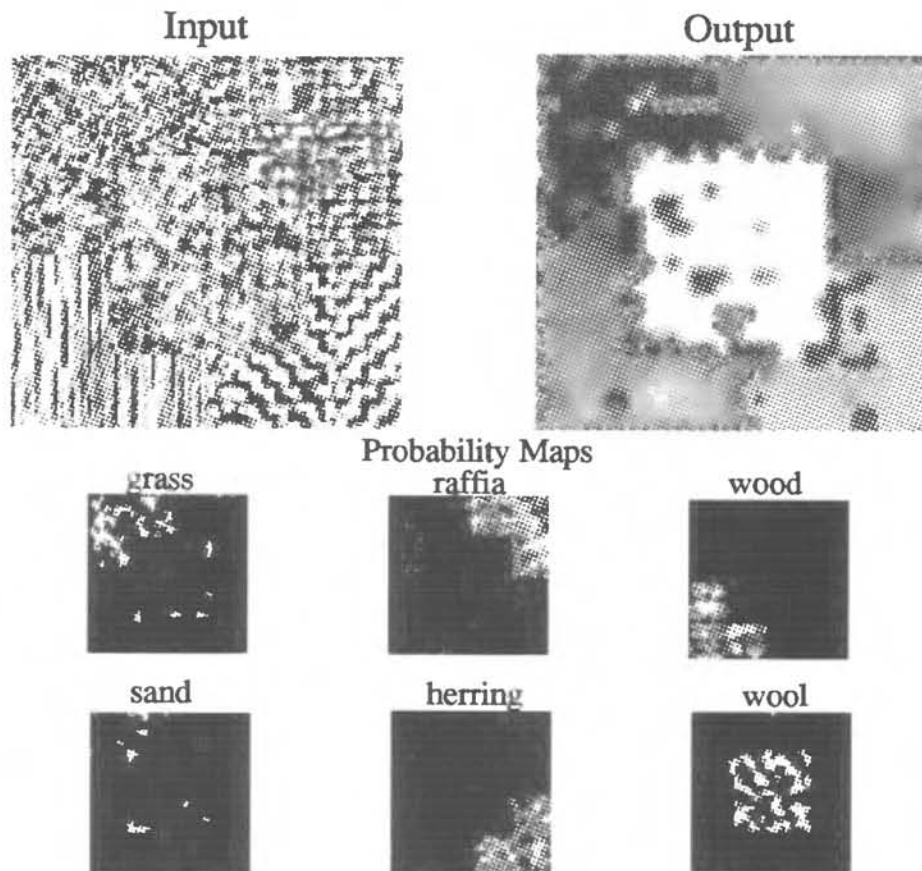

Figure 3: Five class natural texture classification

Input                                    Output

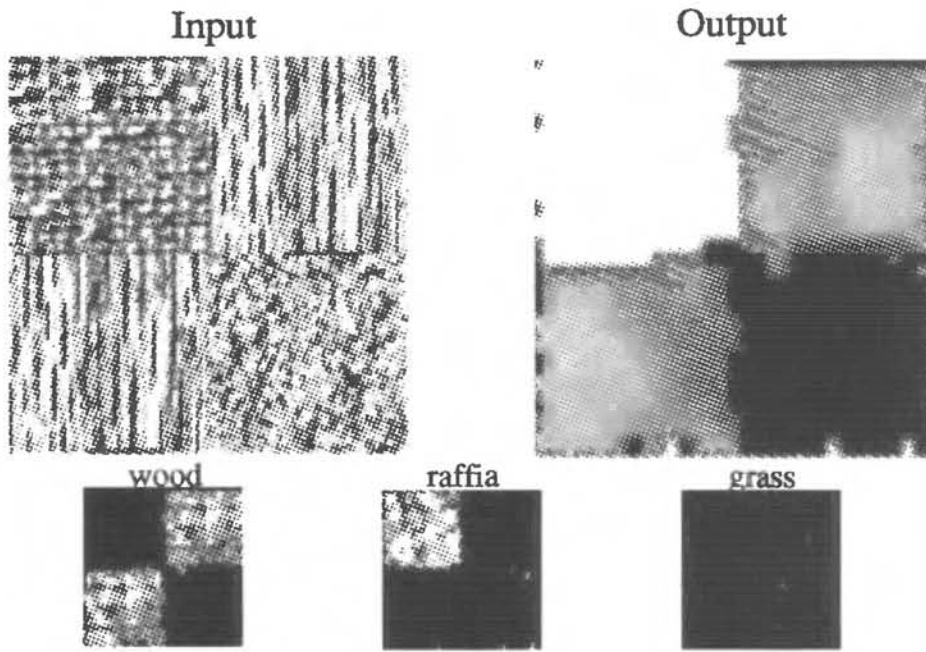

wood              raffia              grass

Figure 4: Identification of an unknown pattern

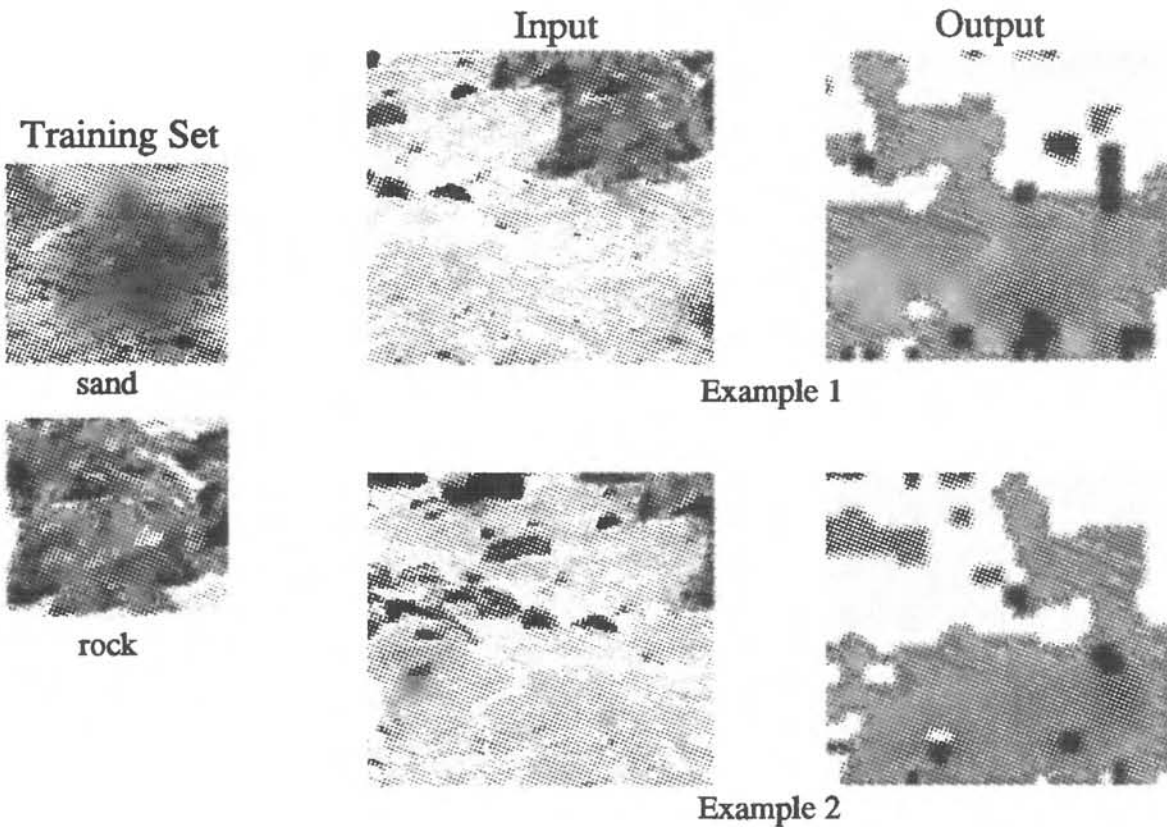

Training Set

sand

rock

Input                                    Output

Example 1

Example 2

Figure 5: Natural scenery analysis

# 6   SUMMARY

The proposed learning scheme achieves a high percentage classification rate on both artificial and natural textures. The combined neural network and rule-based framework enables a probabilistic approach to pattern recognition. In this work we have demonstrated the advantage of utilizing the output probability maps in the pattern recognition process. Complicated patterns were analyzed accurately, with an extension to real-imagery applications. The generalization capability of the system to the discovery of unknown patterns was demonstrated. Future work includes research into scale and rotation invariance capabilities of the presented framework.

## Acknowledgements

This work is funded in part by DARPA under the grant AFOSR-90-0199 and in part by the Army Research Office under the contract DAAL03-89-K-0126. Part of this work was done at Jet Propulsion Laboratory. The advice and software support of the image-analysis group there, especially that of Dr. Charlie Anderson, is greatly appreciated.

## References

H. Greenspan, R. Goodman and R. Chellappa. (1991) Texture Analysis via Unsupervised and Supervised Learning. *Proceedings of the 1991 International Joint Conference on Neural Networks*, Vol. I:639-644.

R. M. Goodman, C. Higgins, J. Miller and P. Smyth. (1992) Rule-Based Networks for Classification and Probability Estimation. to appear in *Neural Computation*.

P. Smyth and R. M. Goodman. (1991) Rule Induction using Information Theory. In G. Piatetsky-Shapiro, W. Frawley (eds.), *Knowledge Discovery in Databases*, 159-176. AAAI Press.

J. Malik and P. Perona. (1990) Preattentive texture discrimination with early vision mechanisms. *Journal of Optical Society of America A*, Vol. 7[5]:923-932.

A. C. Bovik, M. Clark and W. S. Geisler. (1990) Multichannel Texture Analysis Using Localized Spatial Filters. *IEEE Transactions on Pattern Analysis and Machine Intelligence*, 12(1):55-73.

P.J. Burt and E. A. Adelson. (1983) The Laplacian Pyramid as a compact image code. *IEEE Trans. Commun.*,COM-31:532-540.

T. Kohonen. (1984) *Self Organisation and Associative Memory*. Springer-Verlag.